# Optimal Regularized Dual Averaging Methods for Stochastic Optimization

**Xi Chen**
Machine Learning Department
Carnegie Mellon University
xichen@cs.cmu.edu

**Qihang Lin**    **Javier Peña**
Tepper School of Business
Carnegie Mellon University
{qihangl,jfp}@andrew.cmu.edu

## Abstract

This paper considers a wide spectrum of regularized stochastic optimization problems where both the loss function and regularizer can be non-smooth. We develop a novel algorithm based on the regularized dual averaging (RDA) method, that can simultaneously achieve the optimal convergence rates for both convex and strongly convex loss. In particular, for strongly convex loss, it achieves the optimal rate of $O(\frac{1}{N} + \frac{1}{N^2})$ for $N$ iterations, which improves the rate $O(\frac{\log N}{N})$ for previous regularized dual averaging algorithms. In addition, our method constructs the final solution directly from the proximal mapping instead of averaging of all previous iterates. For widely used sparsity-inducing regularizers (e.g., $\ell_1$-norm), it has the advantage of encouraging sparser solutions. We further develop a multi-stage extension using the proposed algorithm as a subroutine, which achieves the uniformly-optimal rate $O(\frac{1}{N} + \exp\{-N\})$ for strongly convex loss.

## 1    Introduction

Many risk minimization problems in machine learning can be formulated into a regularized stochastic optimization problem of the following form:

$$\min_{x \in \mathcal{X}} \{\phi(x) := f(x) + h(x)\}. \tag{1}$$

Here, the set of feasible solutions $\mathcal{X}$ is a convex set in $\mathbb{R}^n$, which is endowed with a norm $\|\cdot\|$ and the dual norm $\|\cdot\|_*$. The regularizer $h(x)$ is assumed to be convex, but could be non-differentiable. Popular examples of $h(x)$ include $\ell_1$-norm and related sparsity-inducing regularizers. The loss function $f(x)$ takes the form: $f(x) := \mathbb{E}_\xi(F(x, \xi)) = \int F(x, \xi) dP(\xi)$, where $\xi$ is a random vector with the distribution $P$. In typical regression or classification tasks, $\xi$ is the input and response (or class label) pair. We assume that for every random vector $\xi$, $F(x, \xi)$ is a convex and continuous function in $x \in \mathcal{X}$. Therefore, $f(x)$ is also convex. Furthermore, we assume that there exist constants $L \geq 0$, $M \geq 0$ and $\widetilde{\mu} \geq 0$ such that

$$\frac{\widetilde{\mu}}{2}\|x - y\|^2 \leq f(y) - f(x) - \langle y - x, f'(x) \rangle \leq \frac{L}{2}\|x - y\|^2 + M\|x - y\|, \quad \forall x, y \in \mathcal{X}, \tag{2}$$

where $f'(x) \in \partial f(x)$, the subdifferential of $f$. We note that this assumption allows us to adopt a wide class of loss functions. For example, if $f(x)$ is smooth and its gradient $f'(x) = \nabla f(x)$ is Lipschitz continuous, we have $L > 0$ and $M = 0$ (e.g., squared or logistic loss). If $f(x)$ is non-smooth but Lipschitz continuous, we have $L = 0$ and $M > 0$ (e.g., hinge loss). If $\widetilde{\mu} > 0$, $f(x)$ is strongly convex and $\widetilde{\mu}$ is the so-called strong convexity parameter.

In general, the optimization problem in Eq.(1) is challenging since the integration in $f(x)$ is computationally intractable for high-dimensional $P$. In many learning problems, we do not even know the underlying distribution $P$ but can only generate *i.i.d.* samples $\xi$ from $P$. A traditional approach is to

consider empirical loss minimization problem where the expectation in $f(x)$ is replaced by its empirical average on a set of training samples $\{\xi_1, \ldots, \xi_m\}$: $f_{\text{emp}}(x) := \frac{1}{m} \sum_{i=1}^{m} F(x, \xi_i)$. However, for modern data-intensive applications, minimization of empirical loss with an off-line optimization solver could suffer from very poor scalability.

In the past few years, many stochastic (sub)gradient methods [6, 5, 8, 12, 14, 10, 9, 11, 7, 18] have been developed to directly solve the stochastic optimization problem in Eq.(1), which enjoy low per-iteration complexity and the capability of scaling up to very large data sets. In particular, at the $t$-th iteration with the current iterate $x_t$, these methods randomly draw a sample $\xi_t$ from $P$; then compute the so-called "stochastic subgradient" $G(x_t, \xi_t) \in \partial_x F(x_t, \xi_t)$ where $\partial_x F(x_t, \xi_t)$ denotes the subdifferential of $F(x, \xi_t)$ with respect to $x$ at $x_t$; and update $x_t$ using $G(x_t, \xi_t)$. These algorithms fall into the class of *stochastic approximation* methods. Recently, Xiao [21] proposed the *regularized dual averaging (RDA)* method and its accelerated version (AC-RDA) based on Nesterov's primal-dual method [17]. Instead of only utilizing a single stochastic subgradient $G(x_t, \xi_t)$ of the current iteration, it updates the parameter vector using the average of all past stochastic subgradients $\{G(x_i, \xi_i)\}_{i=1}^{t}$ and hence leads to improved empirical performances.

In this paper, we propose a novel regularized dual averaging method, called *optimal RDA* or *ORDA*, which achieves the optimal expected convergence rate of $\mathbb{E}[\phi(\widehat{x}) - \phi(x^*)]$, where $\widehat{x}$ is the solution from ORDA and $x^*$ is the optimal solution of Eq.(1). As compared to previous dual averaging methods, it has three main advantages:

1. For strongly convex $f(x)$, ORDA improves the convergence rate of stochastic dual averaging methods $O(\frac{\sigma^2 \log N}{\widetilde{\mu} N}) \approx O(\frac{\log N}{\widetilde{\mu} N})$ [17, 21] to an optimal rate $O\left(\frac{\sigma^2 + M^2}{\widetilde{\mu} N} + \frac{L}{N^2}\right) \approx O\left(\frac{1}{\widetilde{\mu} N}\right)$, where $\sigma^2$ is the variance of the stochastic subgradient, $N$ is the number of iterations, and the parameters $\widetilde{\mu}$, $M$ and $L$ of $f(x)$ are defined in Eq.(2).

2. ORDA is a *self-adaptive* and *optimal* algorithm for solving both convex and strongly convex $f(x)$ with the strong convexity parameter $\widetilde{\mu}$ as an input. When $\widetilde{\mu} = 0$, ORDA reduces to a variant of AC-RDA in [21] with the optimal rate for solving convex $f(x)$. Furthermore, our analysis allows $f(x)$ to be non-smooth while AC-RDA requires the smoothness of $f(x)$. For strongly convex $f(x)$ with $\widetilde{\mu} > 0$, our algorithm achieves the optimal rate of $\left(\frac{1}{\widetilde{\mu} N}\right)$ while AC-RDA does not utilize the advantage of strong convexity.

3. Existing RDA methods [21] and many other stochastic gradient methods (e.g., [14, 10]) can only show the convergence rate for the averaged iterates: $\bar{x}_N = \sum_{t=1}^{N} \varrho_t x_t / \sum_{t=1}^{N} \varrho_t$, where the $\{\varrho_t\}$ are nonnegative weights. However, in general, the average iterates $\bar{x}_N$ cannot keep the structure that the regularizer tends to enforce (e.g., sparsity, low-rank, etc). For example, when $h(x)$ is a sparsity-inducing regularizer ($\ell_1$-norm), although $x_t$ computed from proximal mapping will be sparse as $t$ goes large, the averaged solution could be non-sparse. In contrast, our method directly generates the final solution from the proximal mapping, which leads to sparser solutions.

In addition to the rate of convergence, we also provide *high probability bounds* on the error of objective values. Utilizing a technical lemma from [3], we could show the same high probability bound as in RDA [21] but under a weaker assumption.

Furthermore, using ORDA as a subroutine, we develop the multi-stage ORDA which obtains the convergence rate of $O\left(\frac{\sigma^2 + M^2}{\widetilde{\mu} N} + \exp\{-\sqrt{\widetilde{\mu}/L}N\}\right)$ for strongly convex $f(x)$. Recall that ORDA has the rate $O\left(\frac{\sigma^2 + M^2}{\widetilde{\mu} N} + \frac{L}{N^2}\right)$ for strongly convex $f(x)$. The rate of muli-stage ORDA improves the second term in the rate of ORDA from $O\left(\frac{L}{N^2}\right)$ to $O\left(\exp\{-\sqrt{\widetilde{\mu}/L}N\}\right)$ and achieves the so-called "*uniformly-optimal*" rate [15]. Although the improvement is on the non-dominating term, multi-stage ORDA is an optimal algorithm for both stochastic and deterministic optimization. In particular, for *deterministic* strongly convex and smooth $f(x)$ ($M = 0$), one can use the same algorithm but only replaces the stochastic subgradient $G(x, \xi)$ by the deterministic gradient $\nabla f(x)$. Then, the variance of the stochastic subgradient $\sigma = 0$. Now the term $\frac{\sigma^2 + M^2}{\widetilde{\mu} N}$ in the rate equals to 0 and multi-stage ORDA becomes an optimal deterministic solver with the exponential rate

---

**Algorithm 1** Optimal Regularized Dual Averaging Method: $ORDA(x_0, N, \Gamma, c)$

---

**Input Parameters**: Starting point $x_0 \in \mathcal{X}$, the number of iterations $N$, constants $\Gamma \geq L$ and $c \geq 0$.

**Parameters for** $f(x)$**:** Constants $L$, $M$ and $\widetilde{\mu}$ for $f(x)$ in Eq. (2) and set $\mu = \widetilde{\mu}/\tau$.

**Initialization**: Set $\theta_t = \frac{2}{t+2}$; $\nu_t = \frac{2}{t+1}$; $\gamma_t = c(t+1)^{3/2} + \tau\Gamma$; $z_0 = x_0$.

**Iterate** for $t = 0, 1, 2, \ldots, N$:

1. $y_t = \frac{(1-\theta_t)(\mu+\theta_t^2\gamma_t)}{\theta_t^2\gamma_t+(1-\theta_t^2)\mu} x_t + \frac{(1-\theta_t)\theta_t\mu+\theta_t^3\gamma_t}{\theta_t^2\gamma_t+(1-\theta_t^2)\mu} z_t$

2. Sample $\xi_t$ from the distribution $P(\xi)$ and compute the stochastic subgradient $G(y_t, \xi_t)$.

3. $g_t = \theta_t\nu_t \left( \sum_{i=0}^{t} \frac{G(y_i,\xi_i)}{\nu_i} \right)$

4. $z_{t+1} = \arg\min_{x \in \mathcal{X}} \left\{ \langle x, g_t \rangle + h(x) + \theta_t\nu_t \left( \sum_{i=0}^{t} \frac{\mu V(x,y_i)}{\nu_i} \right) + \theta_t\nu_t\gamma_{t+1}V(x,x_0) \right\}$

5. $x_{t+1} = \arg\min_{x \in \mathcal{X}} \left\{ \langle x, G(y_t, \xi_t) \rangle + h(x) + \left( \frac{\mu}{\tau\theta_t^2} + \frac{\gamma_t}{\tau} \right) V(x,y_t) \right\}$

**Output**: $x_{N+1}$

---

$O\left( \exp\{-\sqrt{\widetilde{\mu}/LN}\} \right)$. This is the reason why such a rate is "uniformly-optimal", i.e., optimal with respect to both stochastic and deterministic optimization.

## 2 Preliminary and Notations

In the framework of first-order stochastic optimization, the only available information of $f(x)$ is the stochastic subgradient. Formally speaking, stochastic subgradient of $f(x)$ at $x$, $G(x, \xi)$, is a vector-valued function such that $\mathbb{E}_\xi G(x, \xi) = f'(x) \in \partial f(x)$. Following the existing literature, a standard assumption on $G(x, \xi)$ is made throughout the paper : there exists a constant $\sigma$ such that $\forall x \in \mathcal{X}$,

$$\mathbb{E}_\xi(\|G(x,\xi) - f'(x)\|_*^2) \leq \sigma^2. \tag{3}$$

A key updating step in dual averaging methods, the proximal mapping, utilizes the Bregman divergence. Let $\omega(x) : \mathcal{X} \to \mathbb{R}$ be a strongly convex and differentiable function, the Bregman divergence associated with $\omega(x)$ is defined as:

$$V(x,y) := \omega(x) - \omega(y) - \langle \nabla\omega(y), x - y \rangle. \tag{4}$$

One typical and simple example is $\omega(x) = \frac{1}{2}\|x\|_2^2$ together with $V(x,y) = \frac{1}{2}\|x - y\|_2^2$. One may refer to [21] for more examples. We can always scale $\omega(x)$ so that $V(x,y) \geq \frac{1}{2}\|x - y\|^2$ for all $x, y \in \mathcal{X}$. Following the assumption in [10]: we assume that $V(x,y)$ *grows quadratically* with the parameter $\tau > 1$, i.e., $V(x,y) \leq \frac{\tau}{2}\|x - y\|^2$ with $\tau > 1$ for all $x, y \in \mathcal{X}$. In fact, we could simply choose $\omega(x)$ with a $\tau$-Lipschitz continuous gradient so that the quadratic growth assumption will be automatically satisfied.

## 3 Optimal Regularized Dual Averaging Method

In dual averaging methods [17, 21], the key proximal mapping step utilizes the average of all past stochastic subgradients to update the parameter vector. In particular, it takes the form: $z_{t+1} = \arg\min_{x \in \mathcal{X}} \left\{ \langle g_t, x \rangle + h(x) + \frac{\beta_t}{t}V(x,x_0) \right\}$, where $\beta_t$ is the step-size and $g_t = \frac{1}{t+1}\sum_{i=0}^{t} G(z_i, \xi_i)$. For strongly convex $f(x)$, the current dual averaging methods achieve a rate of $O(\frac{\sigma^2 \log N}{\widetilde{\mu}N})$, which is suboptimal. In this section, we propose a new dual averaging algorithm which adapts to both strongly and non-strongly convex $f(x)$ via the strong convexity parameter $\widetilde{\mu}$ and achieves optimal rates in both cases. In addition, for previous dual averaging methods, to guarantee the convergence, the final solution takes the form: $\widehat{x} = \frac{1}{N+1}\sum_{t=0}^{N} z_t$ and hence is not sparse in nature for sparsity-inducing regularizers. Instead of taking the average, we introduce another proximal mapping and generate the final solution directly from the second proximal mapping. This strategy will provide us sparser solutions in practice. It is worthy to note that in RDA, $z_N$ has been proved to achieve the desirable sparsity pattern (i.e., manifold identification property) [13]. However, according to [13], the

convergence of $\phi(z_N)$ to the optimal $\phi(x^*)$ is established only under a more restrictive assumption that $x^*$ is a strong local minimizer of $\phi$ relative to the optimal manifold and the convergence rate is quite slow. Without this assumption, the convergence of $\phi(z_N)$ is still unknown.

The proposed optimal RDA (ORDA) method is presented in Algorithm 1. To simplify our notations, we define the parameter $\mu = \widetilde{\mu}/\tau$, which scales the strong convexity parameter $\widetilde{\mu}$ by $\frac{1}{\tau}$, where $\tau$ is the quadratic growth constant. In general, the constant $\Gamma$ which defines the step-size parameter $\gamma_t$ is set to $L$. However, we allow $\Gamma$ to be an arbitrary constant greater than or equal to $L$ to facilitate the introduction of the multi-stage ORDA in the later section. The parameter $c$ is set to achieve the optimal rates for both convex and strongly convex loss. When $\mu > 0$ (or equivalently, $\widetilde{\mu} > 0$), $c$ is set to 0 so that $\gamma_t \equiv \tau\Gamma \geq \tau L$; while for $\mu = 0$, $c = \frac{\sqrt{\tau}(\sigma+M)}{2\sqrt{V(x^*,x_0)}}$. Since $x^*$ is unknown in practice, one might replace $V(x^*, x_0)$ in $c$ by a tuning parameter.

Here, we make a few more explanations of Algorithm 1. In Step 1, the intermediate point $y_t$ is a convex combination of $x_t$ and $z_t$ and when $\mu = 0$, $y_t = (1-\theta_t)x_t + \theta_t z_t$. The choice of the combination weights is inspired by [10]. Second, with our choice of $\theta_t$ and $\nu_t$, it is easy to prove that $\sum_{i=0}^{t} \frac{1}{\nu_i} = \frac{1}{\theta_t \nu_t}$. Therefore, $g_t$ in Step 3 is a convex combination of $\{G(y_i, \xi_i)\}_{i=0}^{t}$. As compared to RDA which uses the average of past subgradients, $g_t$ in ORDA is a *weighted average* of all past stochastic subgradients and the subgradient from the larger iteration has a larger weight (i.e., $G(y_i, \xi_i)$ has the weight $\frac{2(i+1)}{(t+1)(t+2)}$). In practice, instead of storing all past stochastic subgradients, $g_t$ could be simply updated based on $g_{t-1}$: $g_t = \theta_t \nu_t \left( \frac{g_{t-1}}{\theta_{t-1}\nu_{t-1}} + \frac{G(y_t,\xi_t)}{\nu_t} \right)$. We also note that since the error in the stochastic subgradient $G(y_t, \xi_t)$ will affect the sparsity of $x_{t+1}$ via the second proximal mapping, to obtain stable sparsity recovery performances, it would be better to construct the stochastic subgradient with a small batch of samples [21, 1]. This could help to reduce the noise of the stochastic subgradient.

## 3.1 Convergence Rate

We present the convergence rate for ORDA. We start by presenting a general theorem without plugging the values of the parameters. To simplify our notations, we define $\Delta_t := G(y_t, \xi_t) - f'(y_t)$.

**Theorem 1** *For ORDA, if we require $c > 0$ when $\widetilde{\mu} = 0$, then for any $t \geq 0$:*

$$\phi(x_{t+1}) - \phi(x^*) \leq \theta_t \nu_t \gamma_{t+1} V(x^*, x_0) + \frac{\theta_t \nu_t}{2} \sum_{i=0}^{t} \frac{(\|\Delta_i\|_* + M)^2}{\left( \frac{\mu}{\tau\theta_i} + \frac{\theta_i\gamma_i}{\tau} - \theta_i L \right)\nu_i} + \theta_t \nu_t \sum_{i=0}^{t} \frac{\langle x^* - \widehat{z}_i, \Delta_i \rangle}{\nu_i}, \quad (5)$$

*where $\widehat{z}_t = \frac{\theta_t \mu}{\mu + \gamma_t \theta_t^2} y_t + \frac{(1-\theta_t)\mu + \gamma_t \theta_t^2}{\mu + \gamma_t \theta_t^2} z_t$, is a convex combination of $y_t$ and $z_t$; and $\widehat{z}_t = z_t$ when $\mu = 0$. Taking the expectation on both sides of Eq.(5):*

$$\mathbb{E}\phi(x_{t+1}) - \phi(x^*) \leq \theta_t \nu_t \gamma_{t+1} V(x^*, x_0) + (\sigma^2 + M^2)\theta_t \nu_t \sum_{i=0}^{t} \frac{1}{\left( \frac{\mu}{\tau\theta_i} + \frac{\theta_i\gamma_i}{\tau} - \theta_i L \right)\nu_i}. \quad (6)$$

The proof of Theorem 1 is given in Appendix. In the next two corollaries, we establish the rates of convergence in expectation for ORDA by choosing different values for $c$ based on $\widetilde{\mu}$.

**Corollary 1** *For convex $f(x)$ with $\widetilde{\mu} = 0$, by setting $c = \frac{\sqrt{\tau}(\sigma+M)}{2\sqrt{V(x^*,x_0)}}$ and $\Gamma = L$, we obtain:*

$$\mathbb{E}\phi(x_{N+1}) - \phi(x^*) \leq \frac{4\tau L V(x^*, x_0)}{N^2} + \frac{8(\sigma+M)\sqrt{\tau V(x^*,x_0)}}{\sqrt{N}}. \quad (7)$$

Based on Eq.(6), the proof of Corollary 1 is straightforward with the details in Appendix. Since $x^*$ is unknown in practice, one could set $c$ by replacing $V(x^*, x_0)$ in $c$ with any value $D^* \geq V(x^*, x_0)$. By doing so, Eq.(7) remains valid after replacing all $V(x^*, x_0)$ by $D^*$. For convex $f(x)$ with $\widetilde{\mu} = 0$, the rate in Eq.(7) has achieved the *uniformly-optimal* rate according to [15]. In fact, if $f(x)$ is a deterministic and smooth function with $\sigma = M = 0$ (e.g., smooth empirical loss), one only needs

to change the stochastic subgradient $G(y_t, \xi_t)$ to $\nabla f(y_t)$. The resulting algorithm, which reduces to Algorithm 3 in [20], is an optimal deterministic first-order method with the rate $O(\frac{LV(x^*, x_0)}{N^2})$.

We note that the quadratic growth assumption of $V(x, y)$ is not necessary for convex $f(x)$. If one does not assume this assumption and replaces the last step in ORDA by $x_{t+1} = \arg\min_{x \in \mathcal{X}} \left\{ \langle x, G(y_t, \xi_t) \rangle + h(x) + \left( \frac{\mu}{2\theta_t^2} + \frac{\gamma_t}{2} \right) \|x - y_t\|^2 \right\}$, we can achieve the same rate as in Eq.(7) but just removing all $\tau$ from the right hand side. But the quadratic growth assumption is indeed required for showing the convergence for strongly convex $f(x)$ as in the next corollary.

**Corollary 2** *For strongly convex $f(x)$ with $\widetilde{\mu} > 0$, we set $c = 0$ and $\Gamma = L$ and obtain that:*

$$\mathbb{E}\phi(x_{N+1}) - \phi(x^*) \leq \frac{4\tau LV(x^*, x_0)}{N^2} + \frac{4\tau(\sigma^2 + M^2)}{\mu N}. \tag{8}$$

The dominating term in Eq.(8), $O\left(\frac{1}{\mu N}\right)$, is optimal and better than the $O\left(\frac{\log N}{\mu N}\right)$ rate for previous dual averaging methods. However, ORDA has not achieved the uniformly-optimal rate, which takes the form of $O(\frac{\sigma^2 + M^2}{\mu N} + \exp(-\sqrt{\frac{\mu}{L}}N))$. In particular, for deterministic smooth and strongly convex $f(x)$ (i.e., empirical loss with $\sigma = M = 0$), ORDA only achieves the rate of $O(\frac{L}{N^2})$ while the optimal deterministic rate should be $O\left(\exp(-\sqrt{\frac{\mu}{L}}N)\right)$ [16]. Inspired by the multi-restart technique in [7, 11], we present a multi-stage extension of ORDA in Section 4 which achieves the uniformly-optimal convergence rate.

## 3.2 High Probability Bounds

For stochastic optimization problems, another important evaluation criterion is the confidence level of the objective value. In particular, it is of great interest to find $\epsilon(N, \delta)$ as a monotonically decreasing function in both $N$ and $\delta \in (0, 1)$ such that the solution $x_{N+1}$ satisfies $\Pr\left(\phi(x_{N+1}) - \phi(x^*) \geq \epsilon(N, \delta)\right) \leq \delta$. In other words, we want to show that with probability at least $1 - \delta$, $\phi(x_{N+1}) - \phi(x^*) < \epsilon(N, \delta)$. According to Markov inequality, for any $\epsilon > 0$, $\Pr(\phi(x_{N+1}) - \phi(x^*) \geq \epsilon) \leq \frac{\mathbb{E}(\phi(x_{N+1}) - \phi(x^*))}{\epsilon}$. Therefore, we have $\epsilon(N, \delta) = \frac{\mathbb{E}\phi(x_{N+1}) - \phi(x^*)}{\delta}$. Under the basic assumption in Eq.(3), namely $\mathbb{E}_\xi(\|G(x, \xi) - f'(x)\|_*^2) \leq \sigma^2$, and according to Corollary 1 and 2, $\epsilon(N, \delta) = O\left(\frac{(\sigma + M)\sqrt{V(x^*, x_0)}}{\sqrt{N}\delta}\right)$ for convex $f(x)$, and $\epsilon(N, \delta) = O\left(\frac{\sigma^2 + M^2}{\mu N \delta}\right)$ for strongly convex $f(x)$.

However, the above bounds are quite loose. To obtain tighter bounds, we strengthen the basic assumption of the stochastic subgradient in Eq. (3) to the "light-tail" assumption [14]. In particular, we assume that $\mathbb{E}\left(\exp\left\{\|G(x, \xi) - f'(x)\|_*^2/\sigma^2\right\}\right) \leq \exp\{1\}, \ \forall x \in \mathcal{X}$. By further making the boundedness assumption ($\|x^* - \widehat{z}_t\| \leq D$) and utilizing a technical lemma from [3], we obtain a much tighter high probability bound with $\epsilon(N, \delta) = O\left(\frac{\sqrt{\ln(1/\delta)}D\sigma}{\sqrt{N}}\right)$ for both convex and strongly convex $f(x)$. The details are presented in Appendix.

## 4 Multi-stage ORDA for Stochastic Strongly Convex Optimization

As we show in Section 3.1, for convex $f(x)$, ORDA achieves the uniformly-optimal rate. However, for strongly convex $f(x)$, although the dominating term of the convergence rate in Eq.(8) is optimal, the overall rate is not uniformly-optimal. Inspired by the multi-stage stochastic approximation methods [7, 9, 11], we propose the multi-stage extension of ORDA in Algorithm 2 for stochastic strongly convex optimization. For each stage $1 \leq k \leq K$, we run ORDA in Algorithm 1 as a sub-routine for $N_k$ iterations with the parameter $\gamma_t = c(t + 1)^{3/2} + \tau\Gamma$ with $c = 0$ and $\Gamma = \Lambda_k + L$. Roughly speaking, we set $N_k = 2N_{k-1}$ and $\Lambda_k = 4\Lambda_{k-1}$. In other words, we double the number of iterations for the next stage but reduce the step-size. The multi-stage ORDA has achieved uniformly-optimal convergence rate as shown in Theorem 2 with the proof in Appendix. The proof technique follows the one in [11]. Due this specialized proof technique, instead of showing $\mathbb{E}(\phi(x_N)) - \phi(x^*) \leq \epsilon(N)$ as in ORDA, we show the number of iterations $N(\epsilon)$ to achieve the $\epsilon$-accurate solution: $\mathbb{E}(\phi(x_{N(\epsilon)})) - \phi(x^*) \leq \epsilon$. But the two convergence rates are equivalent.

---

**Algorithm 2** Multi-stage ORDA for Stochastic Strongly Convex Optimization

---

**Initialization**: $x_0 \in \mathcal{X}$, a constant $\mathcal{V}_0 \geq \phi(x_0) - \phi(x^*)$ and the number of stages $K$.

**Iterate** for $k = 1, 2, \ldots, K$:

1. Set $N_k = \max \left\{ 4\sqrt{\frac{\tau L}{\mu}}, \frac{2^{k+9}\tau(\sigma^2 + M^2)}{\mu \mathcal{V}_0} \right\}$

2. Set $\Lambda_k = N_k^{3/2} \sqrt{\frac{2^{k-1}\mu(\sigma^2 + M^2)}{\tau \mathcal{V}_0}}$

3. Generate $\widetilde{x}_k$ by calling the sub-routine ORDA($\widetilde{x}_{k-1}, N_k, \Gamma = \Lambda_k + L, c = 0$)

**Output**: $\widetilde{x}_K$

---

**Theorem 2** *If we run multi-stage ORDA for $K$ stages with $K = \log_2\left(\frac{\mathcal{V}_0}{\epsilon}\right)$ for any given $\epsilon$, we have $\mathbb{E}(\phi(\widetilde{x}_K)) - \phi(x^*) \leq \epsilon$ and the total number of iterations is upper bounded by:*

$$N = \sum_{k=1}^{K} N_k \leq 4\sqrt{\frac{\tau L}{\mu}} \log_2\left(\frac{\mathcal{V}_0}{\epsilon}\right) + \frac{1024\tau(\sigma^2 + M^2)}{\mu\epsilon}. \tag{9}$$

## 5 Related Works

In the last few years, a number of stochastic gradient methods [6, 5, 8, 12, 14, 21, 10, 11, 7, 4, 3] have been developed to solve Eq.(1), especially for a sparsity-inducing $h(x)$. In Table 1, we compare the proposed ORDA and its multi-stage extension with some widely used stochastic gradient methods using the following metrics. For the ease of comparison, we assume $f(x)$ is smooth with $M = 0$.

1. The convergence rate for solving (non-strongly) convex $f(x)$ and whether this rate has achieved the uniformly-optimal (Uni-opt) rate.
2. The convergence rate for solving strongly convex $f(x)$ and whether (1) the dominating term of rate is optimal, i.e., $O\left(\frac{\sigma^2}{\mu N}\right)$ and (2) the overall rate is uniformly-optimal.
3. Whether the final solution $\widehat{x}$, on which the results of convergence are built, is generated from the weighted average of previous iterates (Avg) or from the proximal mapping (Prox). For sparsity-inducing regularizers, the solution directly from the proximal mapping is often sparser than the averaged solution.
4. Whether an algorithm allows to use a general Bregman divergence in proximal mapping or it only allows the Euclidean distance $V(x, y) = \frac{1}{2}\|x - y\|_2^2$.

In Table 1, the algorithms in the first 7 rows are stochastic approximation algorithms where only the current stochastic gradient is used at each iteration. The last 4 rows are dual averaging methods where all past subgradients are used. Some algorithms in Table 1 make a more restrictive assumption on the stochastic gradient: $\exists G > 0, \mathbb{E}\|G(x, \xi)\|_*^2 \leq G^2, \forall x \in \mathcal{X}$. It is easy to verify that this assumption implies our basic assumption in Eq.(3) by Jensen's inequality.

As we can see from Table 1, the proposed ORDA possesses all good properties except that the convergence rate for strongly convex $f(x)$ is not uniformly-optimal. Multi-stage ORDA further improves this rate to be uniformly-optimal. In particular, SAGE [8] achieves a nearly optimal rate since the parameter $D$ in the convergence rate is chosen such that $\mathbb{E}\left(\|x_t - x^*\|_2^2\right) \leq D$ for all $t \geq 0$ and it could be much larger than $V \equiv V(x^*, x_0)$. In addition, SAGE requires the boundedness of the domain $\mathcal{X}$, the smoothness of $f(x)$, and only allows the Euclidean distance in proximal mapping. As compared to AC-SA [10] and multi-stage AC-SA [11], our methods do not require the final averaging step; and as shown in our experiments, ORDA has better empirical performances due to the usage of all past stochastic subgradients. Furthermore, we improve the rates of RDA and extend AC-RDA to an optimal algorithm for both convex and strongly convex $f(x)$. Another highly relevant work is [9]. Juditsky et al. [9] proposed multi-stage algorithms to achieve the optimal strongly convex rate based on non-accelerated dual averaging methods. However, the algorithms in [9] assume that $\phi(x)$ is a Lipschitz continuous function, i.e., the subgradient of $\phi(x)$ is bounded. Therefore, when the domain $\mathcal{X}$ is unbounded, the algorithms in [9] cannot be directly applied.

| | Convex $f(x)$ | | Strongly Convex $f(x)$ | | | Final $\hat{x}$ | Bregman |
|---|---|---|---|---|---|---|---|
| | Rate | Uni-opt | Rate | Opt | Uni-opt | | |
| FOBOS [6] | $O\left(\frac{G\sqrt{V}}{\sqrt{N}}\right)$ | NO | $O\left(\frac{G^2\log N}{\bar{\mu}N}\right)$ | NO | NO | Prox | NO |
| COMID [5] | $O\left(\frac{G\sqrt{V}}{\sqrt{N}}\right)$ | NO | $O\left(\frac{G^2\log N}{\bar{\mu}N}\right)$ | NO | NO | Prox | YES |
| SAGE [8] | $O\left(\frac{\sigma\sqrt{D}}{\sqrt{N}}+\frac{LD}{N^2}\right)$ | NEARLY | $O\left(\frac{\sigma^2}{\bar{\mu}N}+\frac{LD}{N^2}\right)$ | YES | NO | Prox | NO |
| AC-SA [10] | $O\left(\frac{\sigma\sqrt{V}}{\sqrt{N}}+\frac{LV}{N^2}\right)$ | YES | $O\left(\frac{\sigma^2}{\bar{\mu}N}+\frac{LV}{N^2}\right)$ | YES | NO | Avg | YES |
| M-AC-SA [11] | NA | NA | $O\left(\frac{\sigma^2}{\bar{\mu}N}+\exp\{-\sqrt{\frac{\bar{\mu}}{L}}N\}\right)$ | YES | YES | Avg | YES |
| Epoch-GD [7] | NA | NA | $O\left(\frac{G^2}{\bar{\mu}N}\right)$ | YES | NO | Avg | NO |
| RDA [21] | $O\left(\frac{G\sqrt{V}}{\sqrt{N}}\right)$ | NO | $O\left(\frac{G^2\log N}{\bar{\mu}N}\right)$ | NO | NO | Avg | YES |
| AC-RDA [21] | $O\left(\frac{\sigma\sqrt{V}}{\sqrt{N}}+\frac{LV}{N^2}\right)$ | YES | NA | NA | NA | Avg | YES |
| **ORDA** | $O\left(\frac{\sigma\sqrt{V}}{\sqrt{N}}+\frac{LV}{N^2}\right)$ | YES | $O\left(\frac{\sigma^2}{\bar{\mu}N}+\frac{LV}{N^2}\right)$ | YES | NO | Prox | YES |
| **M-ORDA** | NA | NA | $O\left(\frac{\sigma^2}{\bar{\mu}N}+\exp\{-\sqrt{\frac{\bar{\mu}}{L}}N\}\right)$ | YES | YES | Prox | YES |

Table 1: Summary for different stochastic gradient algorithms. $V$ is short for $V(x^*, x_0)$; AC for "accelerated"; M for "multi-stage" and NA stands for either "not applicable" or "no analysis of the rate".

Recently, the paper [18] develops another stochastic gradient method which achieves the rate $O(\frac{G^2}{\bar{\mu}N})$ for strongly convex $f(x)$. However, for non-smooth $f(x)$, it requires the averaging of the last a few iterates and this rate is not uniformly-optimal.

## 6    Simulated Experiments

In this section, we conduct simulated experiments to demonstrate the performance of ORDA and its multi-stage extension (M_ORDA). We compare our ORDA and M_ORDA (only for strongly convex loss) with several state-of-the-art stochastic gradient methods, including RDA and AC-RDA [21], AC-SA [10], FOBOS [6] and SAGE [8]. For a fair comparison, we compare all different methods using solutions which have expected convergence guarantees. For all algorithms, we tune the parameter related to step-size (e.g., $c$ in ORDA for convex loss) within an appropriate range and choose the one that leads to the minimum objective value.

In this experiment, we solve a sparse linear regression problem: $\min_{x\in\mathbb{R}^n} f(x)+h(x)$ where $f(x) = \frac{1}{2}\mathbb{E}_{a,b}((a^T x - b)^2) + \frac{\rho}{2}\|x\|_2^2$ and $h(x) = \lambda\|x\|_1$. The input vector $a$ is generated from $N(0, I_{n\times n})$ and the response $b = a^T x^* + \epsilon$, where $x_i^* = 1$ for $1 \le i \le n/2$ and 0 otherwise and the noise $\epsilon \sim N(0,1)$. When $\rho = 0$, th problem is the well known Lasso [19] and when $\rho > 0$, it is known as Elastic-net [22]. The regularization parameter $\lambda$ is tuned so that a deterministic solver on all the samples can correctly recover the underlying sparsity pattern. We set $n = 100$ and create a large pool of samples for generating stochastic gradients and evaluating objective values. The number of iterations $N$ is set to 500. Since we focus on stochastic optimization instead of online learning, we could randomly draw samples from an underlying distribution. So we construct the stochastic gradient using the mini-batch strategy [2, 1] with the batch size 50. We run each algorithm for 100 times and report the mean of the objective value and the F1-score for sparsity recovery performance. F1-score is defined as $2\frac{\text{precision}\cdot\text{recall}}{\text{precision}+\text{recall}}$ where precision $= \sum_{i=1}^p \mathbf{1}_{\{\hat{x}_i=1,x_i^*=1\}}/\sum_{i=1}^p \mathbf{1}_{\{\hat{x}_i=1\}}$ and recall $= \sum_{i=1}^p \mathbf{1}_{\{\hat{x}_i=1,x_i^*=1\}}/\sum_{i=1}^p \mathbf{1}_{\{x_i^*=1\}}$. The higher the F1-score is, the better the recovery ability of the sparsity pattern. The standard deviations for both objective value and the F1-score in 100 runs are very small and thus omitted here due to space limitations.

We first set $\rho = 0$ to test algorithms for (non-strongly) convex $f(x)$. The result is presented in Table 2 (the first two columns). We also plot the decrease of the objective values for the first 200 iterations in Figure 1. From Table 2, ORDA performs the best in both objective value and recovery ability of sparsity pattern. For those optimal algorithms (e.g., AC-RDA, AC-SA, SAGE, ORDA), they achieve lower final objective values and the rates of the decrease are also faster. We note that for dual averaging methods, the solution generated from the (first) proximal mapping (e.g., $z_t$ in

|  | $\rho = 0$ | | $\rho = 1$ | |
|---|---|---|---|---|
|  | Obj | F1 | Obj | F1 |
| RDA | 20.87 | 0.67 | 21.57 | 0.67 |
| AC-RDA | 20.67 | 0.67 | 21.12 | 0.67 |
| AC-SA | 20.66 | 0.67 | 21.01 | 0.67 |
| FOBOS | 20.98 | 0.83 | 21.19 | 0.84 |
| SAGE | 20.65 | 0.82 | 21.09 | 0.73 |
| ORDA | **20.56** | **0.92** | **20.97** | **0.87** |
| M_ORDA | N.A. | N.A. | **20.98** | **0.88** |

Table 2: Comparisons in objective value and F1-score.

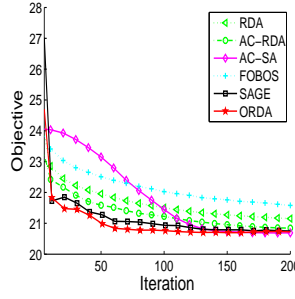

Figure 1: Obj for Lasso.

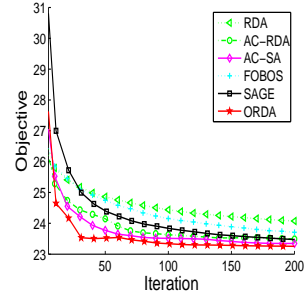

Figure 2: Obj for Elastic-Net.

ORDA) has almost perfect sparsity recovery performance. However, since here is no convergence guarantee for that solution, we do not report results here.

Then we set $\rho = 1$ to test algorithms for solving strongly convex $f(x)$. The results are presented in Table 2 (the last two columns) and Figure 2 and 3. As we can see from Table 2, ORDA and M_ORDA perform the best. Although M_ORDA achieves the theoretical uniformly-optimal convergence rate, the empirical performance of M_ORDA is almost identical to that of ORDA. This observation is consistent with our theoretical analysis since the improvement of the convergence rate only appears on the non-dominating term. In addition, ORDA, M_ORDA, AC-SA and SAGE with the convergence rate $O(\frac{1}{\mu N})$ achieve lower objective values as compared to other algorithms with the rate $O(\frac{\log N}{\mu N})$ . For better visualization, we do

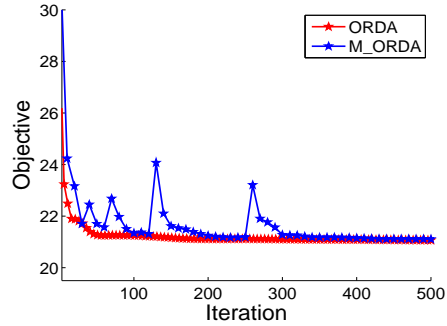

Figure 3: ORDA v.s. M_ORDA.

not include the comparison between M_ORA and ORDA in Figure 2. Instead, we present the comparison separately in Figure 3. From Figure 3, the final objective values of both algorithms are very close. An interesting observation is that, for M_ORDA, each time when a new stage starts, it leads to a sharp increase in the objective value following by a quick drop.

## 7 Conclusions and Future Works

In this paper, we propose a new dual averaging method which achieves the optimal rates for solving stochastic regularized problems with both convex and strongly convex loss functions. We further propose a multi-stage extension to achieve the uniformly-optimal convergence rate for strongly convex loss.

Although we study stochastic optimization problems in this paper, our algorithms can be easily converted into online optimization approaches, where a sequence of decisions $\{x_t\}_{t=1}^{N}$ are generated according to Algorithm 1 or 2. We often measure the quality of an online learning algorithm via the so-called *regret*, defined as $R_N(x^*) = \sum_{t=1}^{N}\big((F(x_t,\xi_t) + h(x_t)) - (F(x^*,\xi_t) + h(x^*))\big)$. Given the expected convergence rate in Corollary 1 and 2, the *expected regret* can be easily derived. For example, for strongly convex $f(x)$: $\mathbb{E}R_N(x^*) \leq \sum_{t=1}^{N}(\mathbb{E}(\phi(x_t)) - \phi(x^*)) \leq \sum_{t=1}^{N} O(\frac{1}{t}) = O(\ln N)$. However, it would be a challenging future work to derive the regret bound for ORDA instead of the expected regret. It would also be interesting to develop the parallel extensions of ORDA (e.g., combining the distributed mini-batch strategy in [21] with ORDA) and apply them to some large-scale real problems.

# References

[1] A. Cotter, O. Shamir, N. Srebro, and K. Sridharan. Better mini-batch algorithms via accelerated gradient methods. In *Advances in Neural Information Processing Systems (NIPS)*, 2011.

[2] O. Dekel, R. Gilad-Bachrach, O. Shamir, and L. Xiao. Optimal distributed online prediction using mini-batches. Technical report, Microsoft Research, 2011.

[3] J. Duchi, P. L. Bartlett, and M. Wainwright. Randomized smoothing for stochastic optimization. arXiv:1103.4296v1, 2011.

[4] J. Duchi, E. Hazan, and Y. Singer. Adaptive subgradient methods for online learning and stochastic optimization. In *Conference on Learning Theory (COLT)*, 2010.

[5] J. Duchi, S. Shalev-Shwartz, Y. Singer, and A. Tewari. Composite objective mirror descent. In *Conference on Learning Theory (COLT)*, 2010.

[6] J. Duchi and Y. Singer. Efficient online and batch learning using forward-backward splitting. *Journal of Machine Learning Research*, 10:2873–2898, 2009.

[7] E. Hazan and S. Kale. Beyond the regret minimization barrier: an optimal algorithm for stochastic strongly-convex optimization. In *Conference on Learning Theory (COLT)*, 2011.

[8] C. Hu, J. T. Kwok, and W. Pan. Accelerated gradient methods for stochastic optimization and online learning. In *Advances in Neural Information Processing Systems (NIPS)*, 2009.

[9] A. Juditsky and Y. Nesterov. Primal-dual subgradient methods for minimizing uniformly convex functions. August 2010.

[10] G. Lan and S. Ghadimi. Optimal stochastic approximation algorithms for strongly convex stochastic composite optimization, part i: a generic algorithmic framework. Technical report, University of Florida, 2010.

[11] G. Lan and S. Ghadimi. Optimal stochastic approximation algorithms for strongly convex stochastic composite optimization, part ii: shrinking procedures and optimal algorithms. Technical report, University of Florida, 2010.

[12] J. Langford, L. Li, and T. Zhang. Sparse online learning via truncated gradient. *Journal of Machine Learning Research*, 10:777–801, 2009.

[13] S. Lee and S. J. Wright. Manifold identification of dual averaging methods for regularized stochastic online learning. In *International Conference on Machine Learning (ICML)*, 2011.

[14] A. Nemirovski, A. Juditsky, G. Lan, and A. Shapiro. Robust stochastic approximation approach to stochastic programming. *SIAM Journal on Optimization*, 19(4):1574–1609, 2009.

[15] A. Nemirovski and D. Yudin. *Problem complexity and method efficiency in optimization*. John Wiley New York, 1983.

[16] Y. Nesterov. *Introductory lectures on convex optimization: a basic course*. Kluwer Academic Pub, 2003.

[17] Y. Nesterov. Primal-dual subgradient methods for convex problems. *Mathematical Programming*, 120:221–259, 2009.

[18] A. Rakhlin, O. Shamir, and K. Sridharan. To average or not to average? making stochastic gradient descent optimal for strongly convex problems. In *International Conference on Machine Learning (ICML)*, 2012.

[19] R. Tibshirani. Regression shrinkage and selection via the lasso. *J.R.Statist.Soc.B*, 58:267–288, 1996.

[20] P. Tseng. On accelerated proximal gradient methods for convex-concave optimization. *SIAM Journal on Optimization (Submitted)*, 2008.

[21] L. Xiao. Dual averaging methods for regularized stochastic learning and online optimization. *Journal of Machine Learning Research*, 11:2543–2596, 2010.

[22] H. Zou and T. Hastie. Regularization and variable selection via the elastic net. *J. R. Statist. Soc. B*, 67(2):301–320, 2005.

